# A Biologically Plausible Algorithm for Reinforcement-shaped Representational Learning

**Maneesh Sahani**
W.M. Keck Foundation Center for Integrative Neuroscience
University of California, San Francisco, CA 94143-0732
maneesh@phy.ucsf.edu

## Abstract

Significant plasticity in sensory cortical representations can be driven in mature animals either by behavioural tasks that pair sensory stimuli with reinforcement, or by electrophysiological experiments that pair sensory input with direct stimulation of neuromodulatory nuclei, but usually not by sensory stimuli presented alone. Biologically motivated theories of representational learning, however, have tended to focus on unsupervised mechanisms, which may play a significant role on evolutionary or developmental timescales, but which neglect this essential role of reinforcement in adult plasticity. By contrast, theoretical reinforcement learning has generally dealt with the acquisition of optimal policies for action in an uncertain world, rather than with the concurrent shaping of sensory representations. This paper develops a framework for representational learning which builds on the relative success of unsupervised generative-modelling accounts of cortical encodings to incorporate the effects of reinforcement in a biologically plausible way.

## 1   Introduction

A remarkable feature of the brain is its ability to adapt to, and learn from, experience. This learning has measurable physiological correlates in terms of changes in the stimulus-response properties of individual neurons in the sensory systems of the brain (as well as in many other areas). While passive exposure to sensory stimuli can have profound effects on the developing sensory cortex, significant plasticity in mature animals tends to be observed only in situations where sensory stimuli are associated with either behavioural or electrical reinforcement. Considerable theoretical attention has been paid to unsupervised learning of representations adapted to natural sensory statistics, and to the learning of optimal policies of action for decision processes; however, relatively little work (particularly of a biological bent) has sought to understand the impact of reinforcement tasks on representation.

To be complete, understanding of sensory plasticity must come at two different levels. At a mechanistic level, it is important to understand how synapses are modified, and how synaptic modifications can lead to observed changes in the response properties of cells. Numerous experiments and models have addressed these questions of *how* sensory plastic-

ity occurs. However, a mechanistic description alone neglects the information-processing aspects of the brain's function. Measured changes in sensory representation must underlie an adaptive change in neural information processing. If we can understand the processing goals of sensory systems, and therefore understand how changes in representation advance these goals in the face of changing experience, we will have shed light on the question of *why* sensory plasticity occurs. This is the goal of the current work.

To approach this goal, we first construct a representational model and associated objective function which together isolate the question of how the reinforcement-related value of a stimulus is learned (the classic problem of reinforcement learning) from the question of how this value impacts the sensory representation. We show that the objective function can be optimised by an expectation-maximisation learning procedure, but suggest that direct optimisation is not biologically plausible, relying as it does on the availability of an exact posterior distribution over the cortical representation given both stimulus and reinforcement-value. We therefore develop and validate (through simulation) an alternative optimisation approach based on the statistical technique of importance sampling.

## 2    Model

The standard algorithms of reinforcement learning (RL) deal with an agent that receives rewards or penalties as it interacts with a world of known structure and, generally Markovian, dynamics [1]. The agent passes through a series of "states", choosing in each one an action which results (perhaps stochastically) in a payoff and in a transition to another state. Associated with each state (or state-action pair) and a given policy of action is a **value**, which represents the expected payoff that would be received if the policy were to be followed starting from that initial state (and initial action). Much work in RL has focused on learning the value function. Often the state that the agent occupies at each point in time is assumed to be directly observable. In other cases, the agent receives only partial information about the state it occupies, although in almost all studies the basic structure of the world is assumed to be known. In these partially observable models, then, the state information (which might be thought of as a form of sensory input) is used to estimate which one of a known group of states is currently occupied, and so a natural representation emerges in terms of a belief-distribution over states.

In the general case, however, the state structure of the world, if indeed a division into discrete states makes sense at all, is unknown. Instead, the agent must simultaneously discover a representation of the sensory inputs suitable for predicting the reinforcement value, and learn the action-contingent value function itself. This general problem is quite difficult. In probabilistic terms, solving it exactly would require coping with a complicated joint distribution over representational structures and value functions. However, using an analogy to the variational inference methods of unsupervised learning [2], we might modularise our approach by factoring this joint into independent distributions over the sensory representation on the one hand and the value function on the other. In this framework approximate estimation might proceed iteratively, using the current value function to tune the sensory representation, and then reëstimating the value function for the revised sensory encoding.

The present work, being concerned with the way in which reinforcement guides sensory representational learning, focuses exclusively on the first of these two steps. Thus, we take the value associated with the current sensory input to be given. This value might represent a current estimate generated in the course of the iterative procedure described above. In many of the reinforcement schedules used in physiological experiments, however, the value is easily determined. For example, in a classical conditioning paradigm the value is independent of action, and is given by the sum of the current reinforcement and the discounted average reinforcement received. Our problem, then, is to develop a biologically plausible algorithm which is able to find a representation of the sensory input which facili-

tates prediction of the value.

Although our eventual goal clearly fits well in the framework of RL, we find it useful to start from a standard theoretical account of *unsupervised* representational learning. The view we adopt fits well with a Helmholtzian account of perceptual processing, in which the sensory cortex interprets the activities of receptors so as to infer the state of the external world that must have given rise to the observed pattern of activation. Perception, by this account, may be thought of as a form of probabilistic inference in a **generative model**. The general structure of such a model involves a set of **latent variables** or **causes** whose values directly reflect the underlying state of the world, along with a parameterisation of effects of these causes on immediate sensory experience. A generative model of visual sensation, for example, might contain a hierarchy of latent variables that, at the top, corresponded to the identities and poses of visual objects or the colour and direction of the illuminating light, and at lower levels, represented local consequences of these more basic causes, for example the orientation and contrast of local edges. Taken together, these variables would provide a causal account for observations that correspond to photoreceptor activation. To apply such a framework as a model for cortical processing, then, we take the sensory cortical activity to represent the inferred values of the latent variables.

Thus, perceptual inference in this framework involves estimating the values of the causal variables that gave rise to the sensory input, while developmental (unsupervised) learning involves discovering the correct causal structure from sensory experience. Such a treatment has been used to account for the structure of simple-cell receptive fields in visual cortex [3, 4], and has been extended to further visual cortical response properties in subsequent studies. In the present work our goal is to consider how such a model might be affected by reinforcement. Thus, in addition to the latent causes $L_i$ that generate a sensory event $S_i$, we consider an associated (possibly action-contingent) value $V_i$. This value is presumably more parsimoniously associated with the *causes* underlying the sensory experience, rather than with the details of the receptor activation, and so we take the sensory input and the corresponding value to be conditionally independent given the cortical representation:

$$\mathsf{P}_\theta(S_i, V_i) = \int dL_i\ \mathsf{P}_\theta(S_i \mid L_i)\mathsf{P}_\theta(V_i \mid L_i)\mathsf{P}_\theta(L_i), \tag{1}$$

where $\theta$ is a general vector of model parameters. Thus, the variables $S_i$, $L_i$ and $V_i$ form a Markov chain. In particular, this means that whatever information $S_i$ carries about $V_i$ is expressed (if the model is well-fit) in the cortical representation $L_i$, making this structure appropriate for value prediction. The causal variables $L_i$ have taken on the rôle of the "state" in standard RL.

## 3 Objective function

The natural objective in reinforcement learning is to maximise some form of accumulated reward. However, the model of (1) is, by itself, descriptive rather than prescriptive. That is, the parameters modelled (those determining the responses in the sensory cortex, rather than in associative or motor areas) do not directly control actions or policies of action. Instead, these descriptive parameters only influence the animal's accumulated reinforcement through the accuracy of the description they generate. As a result, even though the ultimate objective may be to maximise total reward, we need to use objective functions that are closer in spirit to the likelihoods common in probabilistic unsupervised learning.

In particular, we consider functions of the form

$$\mathcal{L}(\theta) = \sum_i \alpha(V_i) \log \mathsf{P}_\theta(S_i) + \beta(V_i) \log \mathsf{P}_\theta(V_i \mid S_i) \tag{2}$$

In this expression, the two log probabilities reflect the accuracy of stimulus representation, and of value prediction, respectively. These two terms would appear alone in a straightforward representational model of the joint distribution over sensory stimuli and values. However, in considering a representational subsystem within a reinforcement learning agent, where the overall goal is to maximise accumulated reward, it seems reasonable that the demand for representative or predictive fidelity depend on the value associated with the stimulus; this dependence is reflected here by a value-based weighting of the log probabilities, which we assume will weight the more valuable cases more heavily.

## 4 Learning

While the objective function (2) does not depend explicitly on the cortical representation variables, it does depend on their distributions, through the marginal likelihoods $P_\theta(S_i) = \int dL_i \; P_\theta(S_i, L_i)$ and $P_\theta(V_i \mid S_i) = \int dL_i \; P_\theta(V_i, L_i \mid S_i)$. For all but the simplest probabilistic models, optimising these integral expressions directly is computationally prohibitive. However, a standard technique called the Expectation-Maximisation (EM) algorithm can be extended in a straightforward way to facilitate optimisation of functions with the form we consider here.

We introduce $2N$ unknown probability distributions over the cortical representation, $Q_\alpha(L_i)$ and $Q_\beta(L_i)$. Then, using Jensen's inequality for convex functions, we obtain a lower bound on the objective function:

$$
\mathcal{L}(\theta) = \sum_i \alpha(V_i) \log \int \frac{Q_\alpha(L_i)}{Q_\alpha(L_i)} P_\theta(S_i, L_i) + \beta(V_i) \log \int \frac{Q_\beta(L_i)}{Q_\beta(L_i)} P_\theta(L_i, V_i \mid S_i)
$$

$$
\geq \sum_i \alpha(V_i) \left( \langle \log P_\theta(S_i, L_i) \rangle_{Q_\alpha(L_i)} + \mathsf{H}[Q_\alpha(L_i)] \right)
$$

$$
+ \beta(V_i) \left( \langle \log P_\theta(L_i, V_i \mid S_i) \rangle_{Q_\beta(L_i)} + \mathsf{H}[Q_\beta(L_i)] \right)
$$

$$
= \mathcal{F}(\theta, Q_\alpha(L_i), Q_\beta(L_i))
$$

It can be shown that, provided both functions are continuous and differentiable, local maxima of the "free-energy" $\mathcal{F}$ with respect to all of its arguments correspond, in their optimal values of $\theta$, to local maxima of $\mathcal{L}$ [5]. Thus, any hill-climbing technique applied to the free-energy functional can be used to find parameters that maximise the objective. In particular, the usual EM approach alternates maximisations (or just steps in the gradient direction) with respect to each of the arguments of $\mathcal{F}$. In our case, this results in the following on-line learning updates made after observing the $i$th data point:

$$
Q_\alpha(L_i) \leftarrow P_\theta(L_i \mid S_i) \tag{3a}
$$
$$
Q_\beta(L_i) \leftarrow P_\theta(L_i \mid V_i, S_i) \tag{3b}
$$
$$
\theta \leftarrow \theta + \eta \nabla_\theta \left( \alpha(V_i) \langle \log P_\theta(S_i, L_i) \rangle_{Q_\alpha(L_i)} + \beta(V_i) \langle \log P_\theta(L_i, V_i \mid S_i) \rangle_{Q_\beta(L_i)} \right) \tag{3c}
$$

where the first two equations represent exact maximisations, while the third is a gradient step, with learning rate $\eta$. It will be useful to rewrite (3c) as

$$
\theta \leftarrow \theta + \eta \left( \alpha(V_i) \langle \nabla_\theta \log P_\theta(S_i, L_i) \rangle_{Q_\alpha(L_i)} + \beta(V_i) \langle \nabla_\theta \log P_\theta(L_i \mid S_i) \rangle_{Q_\beta(L_i)} \right.
$$

$$
\left. + \beta(V_i) \langle \nabla_\theta \log P_\theta(V_i \mid L_i) \rangle_{Q_\beta(L_i)} \right) \tag{3c$'$}
$$

where the conditioning on $S_i$ in the final term in not needed due to the Markovian structure of the model.

## 5 Biologically Plausible Learning

Could something like the updates of (3) underlie the task- or neuromodulator-driven changes that are seen in sensory cortex? Two out of the three steps seem plausible. In (3a), the distribution $P_\theta(L_i \mid S_i)$ represents the animal's beliefs about the latent causes that led to the current sensory experience, and as such is the usual product of perceptual inference. In (3c′), the various log probabilities involved are similarly natural products of perceptual or predictive computations. However, the calculation of the distribution $P_\theta(L_i \mid V_i, S_i)$ in (3b) is less easily reconciled with biological constraints.

There are two difficulties. First, the sensory input, $S_i$, and the information needed to assess its associated value, $V_i$, often arrive at quite different times. However, construction of the posterior distribution in its full detail requires simultaneous knowledge of both $S_i$ and $V_i$, and would therefore only be possible if rich information about the sensory stimulus were to be preserved until the associated value could be determined. The feasibility of such detailed persistence of sensory information is unclear. The second difficulty is an architectural one. The connections from receptor epithelium to sensory areas of cortex are extensive, easily capable of conveying the information needed to estimate $P(L \mid S)$. By contrast, the brain structures that seem to be associated with the evaluation of reinforcement, such as the ventral tegmental area or nucleus basalis, make only sparse projections to early sensory cortex; and these projections are frequently modulatory in character, rather than synaptic. Thus, exact computation of $P(L_i \mid V_i)$ (a component of the full $P(L_i \mid V_i, S_i)$) seems difficult to imagine.

It might seem at first that the former of these two problems would also apply to the weight $\alpha(V_i)$ (in the first term of (3c′)), in that execution of this portion of the update would also need to be delayed until this value-dependent weight could be calculated. On closer examination, however, it becomes evident that this difficulty can be avoided. The trick is that in learning, the weight can be applied to the gradient. Thus, it is sufficient only to remember the gradient, or indeed the corresponding change in synaptic weights. One possible way to do this is to actually carry out an update of the weights when just the sensory stimulus is known, but then consolidate this learning (or not) as indicated by the value-related weight. Such a consolidation signal might easily be carried by a neuromodulatory projection from subcortical nuclei involved in the evaluation of reinforcement.

We propose to solve the problem posed by $P(L \mid S, V)$ in essentially the same way, that is by using information about reinforcement-value to guide modulatory reweighting or consolidation of synaptic changes that are initially based on the sensory stimulus alone. Note that the expectations over $P(L_i \mid S_i, V_i)$ that appear in (3c′) could, in principle, be replaced by sums over samples drawn from the distribution. Since learning is gradual and on-line, such a stochastic gradient ascent algorithm would still converge (in probability) to the optimum. Of course, sampling from this distribution is no more compatible with the foregoing biological constraints than integrating over it. However, consider drawing samples $\tilde{L}_i$ from $P(L_i \mid S_i)$, and then weighting the corresponding terms in the sum by $w(\tilde{L}_i) = P(V_i \mid \tilde{L}_i)/P(V_i \mid S_i)$. Then we have, taking the second term in (3c′) for example,

$$\left\langle \nabla_\theta \log P_\theta(\tilde{L}_i \mid S_i) w(\tilde{L}_i) \right\rangle_{\tilde{L}_i \sim P(L_i \mid S_i)} = \int d\tilde{L}_i \nabla_\theta \log P_\theta(\tilde{L}_i \mid S_i) \frac{P(V_i \mid \tilde{L}_i)}{P(V_i \mid S_i)} P(\tilde{L}_i \mid S_i)$$

$$= \int d\tilde{L}_i \nabla_\theta \log P_\theta(\tilde{L}_i \mid S_i) \frac{P(V_i, \tilde{L}_i \mid S_i)}{P(V_i \mid S_i)} = \left\langle \nabla_\theta \log P_\theta(\tilde{L}_i \mid S_i) \right\rangle_{\tilde{L}_i \sim P(L_i \mid S_i, V_i)}.$$

This approach to learning, which exploits the standard statistical technique of **importance sampling** [6], resolves both of the difficulties discussed above. It implies that reinforcement-related processing and learning in the sensory systems of the brain proceeds in these stages:

1. The sensory input is processed to infer beliefs about the latent causes $\mathsf{P}_\theta(L_i \mid S_i)$. One or more samples $\tilde{L}_i$ are drawn from this distribution.

2. Synaptic weights are updated to follow the gradients $\langle \nabla_\theta \log \mathsf{P}_\theta(S_i, L_i) \rangle_{\mathsf{P}_\theta(L_i \mid S_i)}$ and $\nabla_\theta \log \mathsf{P}_\theta(\tilde{L}_i \mid S_i)$ (corresponding to the first two terms of (3c$'$).

3. The associated value is predicted, both on the basis of the full posterior, giving $\mathsf{P}_\theta(V_i \mid S_i)$, and on the basis of the sample(s), giving $\mathsf{P}_\theta(V_i \mid \tilde{L}_i)$.

4. The actual value is observed or estimated, facilitating calculation of the weights $\alpha(V_i)$, $\beta(V_i)$, and $w(\tilde{L}_i)$.

5. These weights are conveyed to sensory cortex and used to consolidate (or not) the synaptic changes of step 2.

This description does not encompass the updates corresponding to the third term of (3c$'$). Such updates could be undertaken once the associated value became apparent; however, the parameters that represent the explicit dependence of value on the latent variables are unlikely to lie in the sensory cortex itself (instead determining computations in subsequent processing).

### 5.1 Distributional Sampling

A commonly encountered difficulty with importance sampling has to do with the distribution of importance weights $w_i$. If the range of weights is too extensive, the optimisation will be driven primarily by few large weights, leading to slow and noisy learning. Fortunately, it is possible to formulate an alternative, in which *distributions* over the cortical representational variables, rather than samples of the variables themselves, are randomly generated and weighted appropriately.[1]

Let $\widetilde{\mathsf{P}}_i(L)$ be a distribution over the latent causes $L$, drawn randomly from a functional distribution $\mathcal{P}(\widetilde{\mathsf{P}}_i \mid S_i)$, such that $\left\langle \widetilde{\mathsf{P}}_i(L) \right\rangle_{\mathcal{P}(\widetilde{\mathsf{P}}_i \mid S_i)} = \mathsf{P}(L_i \mid S_i)$. Then, by analogy with the result above, it can be shown that given importance weights

$$w(\widetilde{\mathsf{P}}_i) = \frac{\int dL\, \mathsf{P}(V_i \mid L) \widetilde{\mathsf{P}}_i(L)}{\mathsf{P}(V_i \mid S_i)}, \tag{4}$$

we have

$$\left\langle \left\langle \nabla_\theta \log \mathsf{P}_\theta(\tilde{L}_i \mid S_i) \right\rangle_{\widetilde{\mathsf{P}}_i(L)} w(\widetilde{\mathsf{P}}_i) \right\rangle_{\widetilde{\mathsf{P}}_i \sim \mathcal{P}(\widetilde{\mathsf{P}}_i \mid S_i)} = \left\langle \nabla_\theta \log \mathsf{P}_\theta(L_i \mid S_i) \right\rangle_{\tilde{L}_i \sim \mathsf{P}(L_i \mid S_i, V_i)}.$$

These distributional samples can thus be used in almost exactly the same manner as the single-valued samples described above.

## 6 Simulation

A paradigmatic generative model structure is that underlying factor analysis (FA) [7], in which both latent and observed variables are normally distributed:

$$\mathsf{P}_\theta(L_i) = \mathcal{N}(0, I)\,; \;\; \mathsf{P}_\theta(S_i \mid L_i) = \mathcal{N}(\Lambda_S L_i, \Psi_S)\,; \;\; \mathsf{P}_\theta(V_i \mid L_i) = \mathcal{N}(\Lambda_V L_i, \Psi_V)\,. \tag{5}$$

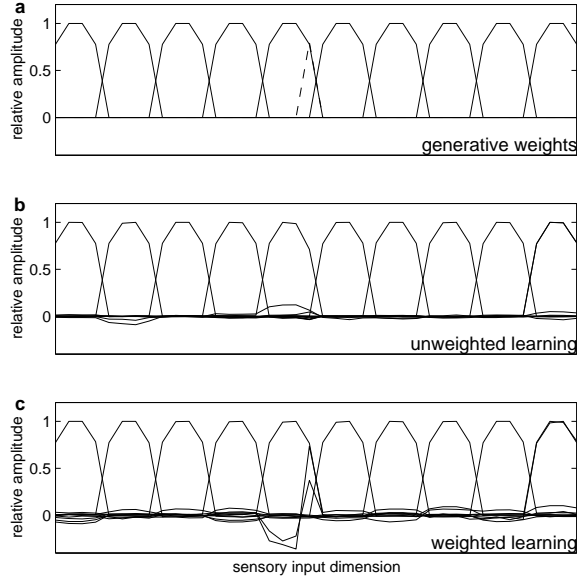

Figure 1: Generative and learned sensory weights. See text for details.

The parameters of the FA model (grouped here in $\theta$) comprise two linear weight matrices $\Lambda_S$ and $\Lambda_V$ and two diagonal noise covariance matrices $\Psi_S$ and $\Psi_V$. This model is similar in its linear generative structure to the independent components analysis models that have previously been employed in accounts of unsupervised development of visual cortical properties [3, 4]; the only difference is in the assumed distribution of the latent variables. The unit normal assumption of FA introduces a rotational degeneracy in solutions. This can be resolved in general by constraining the weight matrix $\Lambda = [\Lambda_S, \Lambda_V]$ to be orthogonal – giving a version of FA known as principal factor analysis (PFA).

We used a PFA-based simulation to verify that the distributional importance-weighted sampling procedure described here is indeed able to learn the correct model given sensory and reinforcement-value data. Random vectors representing sensory inputs and associated values were generated according to (5); these were then used as inputs to a learning system. The objective function optimised had both value-dependent weights $\alpha(V_i)$ and $\beta(V_i)$ set to unity; thus the learning system simply attempted to model the joint distribution of sensory and reinforcement data.

The generative model comprised 11 latent variables, 40 observed sensory variables (which were arranged linearly so as to represent 40 discrete values along a single sensory axis), and a single reinforcement variable. Ten of the latent variables only affected the sensory observations. The weight vectors corresponding to each of these are shown by the solid lines in figure 1a. These "tuning curves" were designed to be orthogonal. The curves shown in figure 1a have been rescaled to have equal maximal amplitude; in fact the amplitudes were randomly varied so that they formed a unique orthogonal basis for the data. These features of the generative weight matrix were essential for PFA to be able to recover the generative model uniquely. The final latent variable affected both reinforcement value and the sensory input at a single point (indicated by the dashed line in figure 1a). Since the output noise matrix in PFA can associate arbitrary variance with each sensory variable, a model fit to only the sensory data would treat the influence of this latent cause as noise. Only when the joint distribution over both sensory input and reinforcement is modelled

will this aspect of the sensory data be captured in the model parameters.

Learning was carried out by processing data generated by the model described above one sample at a time. The posterior distribution $\mathsf{P}_\theta(L_i \mid S_i)$ for the PFA model is Gaussian, with covariance $\Sigma_L = (I + \Lambda_S^\mathsf{T} \Psi_S^{-1} \Lambda_S)^{-1}$ and mean $\mu_L = \Sigma_L \Lambda_S^\mathsf{T} \Psi_S^{-1} S_i$. The distributional samples $\widetilde{\mathsf{P}}_i$ were also taken to be Gaussian. Each had covariance $0.6\Sigma_L$ and mean drawn randomly from $\mathcal{N}(\mu_L, 0.4\Sigma_L)$.

Two simulations were performed. In one case learning proceeded according to the sampled distributions $\widetilde{\mathsf{P}}_i$, with no importance weighting. In the other, learning was modulated by the importance weights given by (4). In all other regards the two simulations were identical. In particular, in both cases the reinforcement predictive weights $\Lambda_V$ were estimated, and in both cases the orthogonality constraint of PFA was applied to the combined estimated weight matrix $[\Lambda_S, \Lambda_V]$. Figure 1b and c shows the sensory weights $\Lambda_S$ learnt by each of these procedures (again the curves have been rescaled to show relative weights). Both algorithms recovered the basic tuning properties; however, only the importance sampling algorithm was able to model the additional data feature that was linked to the prediction of reinforcement value. The fact that in all other regards the two learning simulations were identical demonstrates that the importance weighting procedure (rather than, say, the orthogonality constraint) was responsible for this difference.

## 7   Summary

This paper has presented a framework within which the experimentally observed impact of behavioural reinforcement on sensory plasticity might be understood. This framework rests on a similar foundation to the recent work that has related unsupervised learning to sensory response properties. It extends this foundation to consider prediction of the reinforcement value associated with sensory stimuli. Direct learning by expectation-maximisation within this framework poses difficulties regarding biological plausibility. However, these were resolved by the introduction of an importance sampled approach, along with its extension to distributional sampling. Information about reinforcement is thus carried by a weighting signal that might be identified with the neuromodulatory signals in the brain.

## Footnotes

[1]This sampling scheme can also be formalised as standard importance sampling carried out with a cortical representation re-expressed in terms of the parameters determining the distribution $\widetilde{\mathsf{P}}_i(L)$.

### References

[1]  R. S. Sutton and A. G. Barto. *Reinforcement Learning: An Introduction*. MIT Press, Cambridge, MA, 1998.

[2]  M. I. Jordan, Z. Ghahramani, T. Jaakkola, and L. K. Saul. An introduction to variational methods for graphical models. *Mach. Learning*, 37(2):183–233, 1999.

[3]  B. A. Olshausen and D. J. Field. Emergence of simple-cell receptive field properties by learning a sparse code for natural images. *Nature*, 381(6583):607–9, 1996.

[4]  A. J. Bell and T. J. Sejnowski. The "independent components" of natural scenes are edge filters. *Vision Res.*, 37(23):3327–3338, 1997.

[5]  R. M. Neal and G. E. Hinton. A view of the EM algorithm that justifies incremental, sparse, and other variants. In M. I. Jordan, ed., *Learning in Graphical Models*, pp. 355–370. Kluwer Academic Press, 1998.

[6]  W. H. Press, S. A. Teukolsky, W. T. Vetterling, and B. P. Flannery. *Numerical Recipes in C: The Art of Scientific Computing*. CUP, Cambridge, 2nd edition, 1993.

[7]  B. S. Everitt. *An Introduction to Latent Variable Models*. Chapman and Hall, London, 1984.
